# Probability Estimates for Multi-class Classification by Pairwise Coupling

**Ting-Fan Wu     Chih-Jen Lin**
Department of Computer Science
National Taiwan University
Taipei 106, Taiwan

**Ruby C. Weng**
Department of Statistics
National Chenechi University
Taipei 116, Taiwan

## Abstract

Pairwise coupling is a popular multi-class classification method that combines together all pairwise comparisons for each pair of classes. This paper presents two approaches for obtaining class probabilities. Both methods can be reduced to linear systems and are easy to implement. We show conceptually and experimentally that the proposed approaches are more stable than two existing popular methods: voting and [3].

## 1   Introduction

The multi-class classification problem refers to assigning each of the observations into one of $k$ classes. As two-class problems are much easier to solve, many authors propose to use two-class classifiers for multi-class classification. In this paper we focus on techniques that provide a multi-class classification solution by combining all pairwise comparisons.

A common way to combine pairwise comparisons is by voting [6, 2]. It constructs a rule for discriminating between every pair of classes and then selecting the class with the most winning two-class decisions. Though the voting procedure requires just pairwise decisions, it only predicts a class label. In many scenarios, however, probability estimates are desired. As numerous (pairwise) classifiers do provide class probabilities, several authors [12, 11, 3] have proposed probability estimates by combining the pairwise class probabilities.

Given the observation $\mathbf{x}$ and the class label $y$, we assume that the estimated pairwise class probabilities $r_{ij}$ of $\mu_{ij} = p(y = i \mid y = i \text{ or } j, \mathbf{x})$ are available. Here $r_{ij}$ are obtained by some binary classifiers. Then, the goal is to estimate $\{p_i\}_{i=1}^k$, where $p_i = p(y = i \mid \mathbf{x}), i = 1, \ldots, k$. We propose to obtain an approximate solution to an identity, and then select the label with the highest estimated class probability. The existence of the solution is guaranteed by theory in finite Markov Chains. Motivated by the optimization formulation of this method, we propose a second approach. Interestingly, it can also be regarded as an improved version of the coupling approach given by [12]. Both of the proposed methods can be reduced to solving linear systems and are simple in practical implementation. Furthermore, from conceptual and experimental points of view, we show that the two proposed methods are more stable than voting and the method in [3].

We organize the paper as follows. In Section 2, we review two existing methods. Sections 3 and 4 detail the two proposed approaches. Section 5 presents the relationship among the four methods through their corresponding optimization formulas. In Section 6, we compare

these methods using simulated and real data. The classifiers considered are support vector machines. Section 7 concludes the paper. Due to space limit, we omit all detailed proofs. A complete version of this work is available at `http://www.csie.ntu.edu.tw/~cjlin/papers/svmprob/svmprob.pdf`.

## 2 Review of Two Methods

Let $r_{ij}$ be the estimates of $\mu_{ij} = p_i/(p_i + p_j)$. The voting rule [6, 2] is

$$\delta_V = \text{argmax}_i [\sum_{j:j\neq i} I_{\{r_{ij} > r_{ji}\}}]. \tag{1}$$

A simple estimate of probabilities can be derived as $p_i^v = 2 \sum_{j:j\neq i} I_{\{r_{ij} > r_{ji}\}}/(k(k-1))$. The authors of [3] suggest another method to estimate class probabilities, and they claim that the resulting classification rule can outperform $\delta_V$ in some situations. Their approach is based on the minimization of the Kullback-Leibler (KL) distance between $r_{ij}$ and $\mu_{ij}$:

$$l(\mathbf{p}) = \sum_{i\neq j} n_{ij} r_{ij} \log(r_{ij}/\mu_{ij}), \tag{2}$$

where $\sum_{i=1}^{k} p_i = 1, p_i > 0, i = 1, \ldots, k$, and $n_{ij}$ is the number of instances in class $i$ or $j$. By letting $\nabla l(\mathbf{p}) = 0$, a nonlinear system has to be solved. [3] proposes an iterative procedure to find the minimum of (2). If $r_{ij} > 0, \forall i \neq j$, the existence of a unique global minimal solution to (2) has been proved in [5] and references therein. Let $\mathbf{p}^*$ denote this point. Then the resulting classification rule is

$$\delta_{HT}(x) = \text{argmax}_i [p_i^*].$$

It is shown in Theorem 1 of [3] that

$$p_i^* > p_j^* \text{ if and only if } \tilde{p}_i > \tilde{p}_j, \text{ where } \tilde{p}_j = \frac{2\sum_{s:s\neq j} r_{js}}{k(k-1)}; \tag{3}$$

that is, the $\tilde{p}_i$ are in the same order as the $p_i^*$. Therefore, $\tilde{\mathbf{p}}$ are sufficient if one only requires the classification rule. In fact, as pointed out by [3], $\tilde{\mathbf{p}}$ can be derived as an approximation to the identity by replacing $p_i + p_j$ with $2/k$, and $\mu_{ij}$ with $r_{ij}$.

$$p_i = \sum_{j:j\neq i} (\frac{p_i + p_j}{k-1})(\frac{p_i}{p_i + p_j}) = \sum_{j:j\neq i} (\frac{p_i + p_j}{k-1})\mu_{ij} \tag{4}$$

## 3 Our First Approach

Note that $\delta_{HT}$ is essentially $\text{argmax}_i [\tilde{p}_i]$, and $\tilde{\mathbf{p}}$ is an approximate solution to (4). Instead of replacing $p_i + p_j$ by $2/k$, in this section we propose to solve the system:

$$p_i = \sum_{j:j\neq i} (\frac{p_i + p_j}{k-1}) r_{ij}, \forall i, \qquad \text{subject to } \sum_{i=1}^{k} p_i = 1, p_i \geq 0, \forall i. \tag{5}$$

Let $\bar{\mathbf{p}}$ denote the solution to (5). Then the resulting decision rule is

$$\delta_1 = \text{argmax}_i [\bar{p}_i].$$

As $\delta_{HT}$ relies on $p_i + p_j \approx k/2$, in Section 6.1 we use two examples to illustrate possible problems with this rule.

To solve (5), we rewrite it as

$$Q\mathbf{p} = \mathbf{p}, \quad \sum_{i=1}^{k} p_i = 1, \quad p_i \geq 0, \forall i, \text{ where } Q_{ij} = \begin{cases} r_{ij}/(k-1) & \text{if } i \neq j, \\ \sum_{s:s\neq i} r_{is}/(k-1) & \text{if } i = j. \end{cases} \quad (6)$$

Observe that $\sum_{j=1}^{k} Q_{ij} = 1$ for $i = 1, \ldots, k$ and $0 \leq Q_{ij} \leq 1$ for $i, j = 1, \ldots, k$, so there exists a finite Markov Chain whose transition matrix is $Q$. Moreover, if $r_{ij} > 0$ for all $i \neq j$, then $Q_{ij} > 0$, which implies this Markov Chain is irreducible and aperiodic. These conditions guarantee the existence of a unique stationary probability and all states being positive recurrent. Hence, we have the following theorem:

**Theorem 1** *If $r_{ij} > 0$, $i \neq j$, then (6) has a unique solution $\mathbf{p}$ with $0 < p_i < 1$, $\forall i$.*

With Theorem 1 and some further analyses, if we remove the constraint $p_i \geq 0, \forall i$, the linear system with $k + 1$ equations still has the same unique solution. Furthermore, if any one of the $k$ equalities $Q\mathbf{p} = \mathbf{p}$ is removed, we have a system with $k$ variables and $k$ equalities, which, again, has the same single solution. Thus, (6) can be solved by Gaussian elimination. On the other hand, as the stationary solution of a Markov Chain can be derived by the limit of the $n$-step transition probability matrix $Q^n$, we can solve $\mathbf{p}$ by repeatedly multiplying $Q^T$ with any initial vector.

Now we reexamine this method to gain more insight. The following arguments show that the solution to (5) is a global minimum of a meaningful optimization problem. To begin, we express (5) as $\sum_{j:j\neq i} r_{ji}p_i - \sum_{j:j\neq i} r_{ij}p_j = 0, i = 1, \ldots, k$, using the property that $r_{ij} + r_{ji} = 1, \forall i \neq j$. Then the solution to (5) is in fact the global minimum of the following problem:

$$\min_{\mathbf{p}} \sum_{i=1}^{k} \left( \sum_{j:j\neq i} r_{ji}p_i - \sum_{j:j\neq i} r_{ij}p_j \right)^2 \quad \text{subject to } \sum_{i=1}^{k} p_i = 1, p_i \geq 0, \forall i. \quad (7)$$

Since the object function is always nonnegative, and it attains zero under (5) and (6).

## 4 Our Second Approach

Note that both approaches in Sections 2 and 3 involve solving optimization problems using the relations like $p_i/(p_i + p_j) \approx r_{ij}$ or $\sum_{j:j\neq i} r_{ji}p_i \approx \sum_{j:j\neq i} r_{ij}p_j$. Motivated by (7), we suggest another optimization formulation as follows:

$$\min_{\mathbf{p}} \frac{1}{2} \sum_{i=1}^{k} \sum_{j:j\neq i} (r_{ji}p_i - r_{ij}p_j)^2 \quad \text{subject to } \sum_{i=1}^{k} p_i = 1, p_i \geq 0, \forall i. \quad (8)$$

In related work, [12] proposes to solve a linear system consisting of $\sum_{i=1}^{k} p_i = 1$ and *any* $k - 1$ equations of the form $r_{ji}p_i = r_{ij}p_j$. However, pointed out in [11], the results of [12] strongly depends on the selection of $k - 1$ equations. In fact, as (8) considers all $r_{ij}p_j - r_{ji}p_i$, not just $k - 1$ of them, it can be viewed as an improved version of [12].

Let $\mathbf{p}^\dagger$ denote the corresponding solution. We then define the classification rule as

$$\delta_2 = \text{argmax}_i[p_i^\dagger].$$

Since (7) has a unique solution, which can be obtained by solving a simple linear system, it is desired to see whether the minimization problem (8) has these nice properties. In the rest of the section, we show that this is true. The following theorem shows that the nonnegative constraints in (8) are redundant.

**Theorem 2** *Problem* (8) *is equivalent to a simplification without conditions* $p_i \geq 0, \forall i.$

Note that we can rewrite the objective function of (8) as

$$\min_{\mathbf{p}} = \frac{1}{2}\mathbf{p}^T Q \mathbf{p}, \qquad \text{where } Q_{ij} = \begin{cases} \sum_{s:s \neq i} r_{si}^2 & \text{if } i = j, \\ r_{ji}r_{ij} & \text{if } i \neq j. \end{cases} \qquad (9)$$

From here we can show that $Q$ is positive semi-definite. Therefore, without constraints $p_i \geq 0, \forall i$, (9) is a linear-constrained convex quadratic programming problem. Consequently, a point $\mathbf{p}$ is a global minimum if and only if it satisfies the KKT optimality condition: There is a scalar $b$ such that

$$\begin{bmatrix} Q & \mathbf{e} \\ \mathbf{e}^T & 0 \end{bmatrix} \begin{bmatrix} \mathbf{p} \\ b \end{bmatrix} = \begin{bmatrix} 0 \\ 1 \end{bmatrix}. \qquad (10)$$

Here $\mathbf{e}$ is the vector of all ones and $b$ is the Lagrangian multiplier of the equality constraint $\sum_{i=1}^{k} p_i = 1$. Thus, the solution of (8) can be obtained by solving the simple linear system (10). The existence of a unique solution is guaranteed by the invertibility of the matrix of (10). Moreover, if $Q$ is positive definite(PD), this matrix is invertible. The following theorem shows that $Q$ is PD under quite general conditions.

**Theorem 3** *If for any* $i = 1, \ldots, k$, *there are* $s \neq i$ *and* $j \neq i$ *such that* $\frac{r_{si}r_{sj}}{r_{is}} \neq \frac{r_{ji}r_{js}}{r_{ij}}$, *then* $Q$ *is positive definite.*

In addition to direct methods, next we propose a simple iterative method for solving (10):

**Algorithm 1**

1. *Start with some initial* $p_i \geq 0, \forall i$ *and* $\sum_{i=1}^{k} p_i = 1.$

2. *Repeat* $(t = 1, \ldots, k, 1, \ldots)$

$$p_t \leftarrow \frac{1}{Q_{tt}}[-\sum_{j:j \neq t} Q_{tj}p_j + \mathbf{p}^T Q \mathbf{p}] \qquad (11)$$

$$normalize \ \mathbf{p} \qquad (12)$$

*until* (10) *is satisfied.*

**Theorem 4** *If* $r_{sj} > 0, \forall s \neq j$, *and* $\{\mathbf{p}^i\}_{i=1}^{\infty}$ *is the sequence generated by Algorithm 1, any convergent sub-sequence goes to a global minimum of (8).*

As Theorem 3 indicates that in general $Q$ is positive definite, the sequence $\{\mathbf{p}^i\}_{i=1}^{\infty}$ from Algorithm 1 usually globally converges to the unique minimum of (8).

## 5   Relations Among Four Methods

The four decision rules $\delta_{HT}$, $\delta_1$, $\delta_2$, and $\delta_V$ can be written as $\text{argmax}_i[p_i]$, where $\mathbf{p}$ is derived by the following four optimization formulations under the constants $\sum_{i=1}^{k} p_i = 1$

and $p_i \geq 0, \forall i$:

$$\delta_{HT} : \min_{\mathbf{p}} \sum_{i=1}^{k} [\sum_{j:j\neq i}^{k} (r_{ij}\frac{1}{k} - \frac{1}{2}p_i)]^2, \tag{13}$$

$$\delta_1 : \quad \min_{\mathbf{p}} \sum_{i=1}^{k} [\sum_{j:j\neq i}^{k} (r_{ij}p_j - r_{ji}p_i)]^2, \tag{14}$$

$$\delta_2 : \quad \min_{\mathbf{p}} \sum_{i=1}^{k} \sum_{j:j\neq i}^{k} (r_{ij}p_j - r_{ji}p_i)^2, \tag{15}$$

$$\delta_V : \quad \min_{\mathbf{p}} \sum_{i=1}^{k} \sum_{j:j\neq i}^{k} (I_{\{r_{ij}>r_{ji}\}}p_j - I_{\{r_{ji}>r_{ij}\}}p_i)^2. \tag{16}$$

Note that (13) can be easily verified, and that (14) and (15) have been explained in Sections 3 and 4. For (16), its solution is

$$p_i = \frac{c}{\sum_{j:j\neq i} I_{\{r_{ji}>r_{ij}\}}},$$

where $c$ is the normalizing constant;* and therefore, $\text{argmax}_i[p_i]$ is the same as (1). Clearly, (13) can be obtained from (14) by letting $p_j \approx 1/k, \forall j$ and $r_{ji} \approx 1/2, \forall i, j$. Such approximations ignore the differences between $p_i$. Similarly, (16) is from (15) by taking the extreme values of $r_{ij}$: 0 or 1. As a result, (16) may enlarge the differences between $p_i$. Next, compared with (15), (14) may tend to underestimate the differences between the $p_i$'s. The reason is that (14) allows the difference between $r_{ij}p_j$ and $r_{ji}p_i$ to get canceled first. Thus, conceptually, (13) and (16) are more extreme – the former tends to underestimate the differences between $p_i$'s, while the latter overestimate them. These arguments will be supported by simulated and real data in the next section.

## 6 Experiments

### 6.1 Simple Simulated Examples

[3] designs a simple experiment in which all $p_i$'s are fairly close and their method $\delta_{HT}$ outperforms the voting strategy $\delta_V$. We conduct this experiment first to assess the performance of our proposed methods. As in [3], we define class probabilities $p_1 = 1.5/k$, $p_j = (1 - p_1)/(k - 1)$, $j = 2, \ldots, k$, and then set

$$r_{ij} = \frac{p_i}{p_i + p_j} + 0.1z_{ij} \text{ if } i > j, \tag{17}$$

$$r_{ji} = 1 - r_{ij} \qquad \text{if } j > i, \tag{18}$$

where $z_{ij}$ are standard normal variates. Since $r_{ij}$ are required to be within (0,1), we truncate $r_{ij}$ at $\epsilon$ below and $1 - \epsilon$ above, with $\epsilon = 0.00001$. In this example, class 1 has the highest probability and hence is the correct class.

Figure 1 shows accuracy rates for each of the four methods when $k = 3, 5, 8, 10, 12, 15, 20$. The accuracy rates are averaged over 1,000 replicates. Note that in this experiment all classes are quite competitive, so, when using $\delta_V$, sometimes the highest vote occurs at two

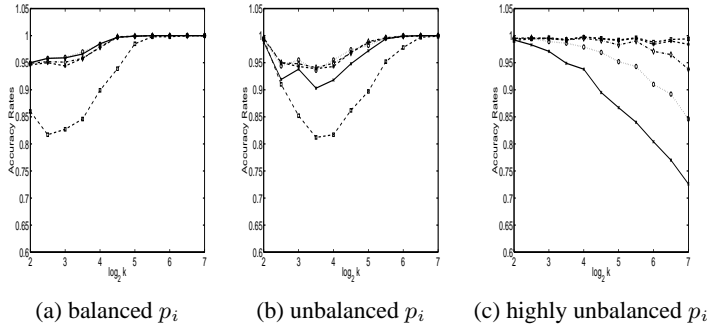

| (a) balanced $p_i$ | (b) unbalanced $p_i$ | (c) highly unbalanced $p_i$ |

Figure 1: Accuracy of predicting the true class by the methods $\delta_{HT}$ (solid line, cross marked), $\delta_V$ (dash line, square marked), $\delta_1$ (dotted line, circle marked), and $\delta_2$ (dashed line, asterisk marked) from simulated class probability $p_i, i = 1, 2 \cdots k$.

or more different classes. We handle this problem by randomly selecting one class from the ties. This partly explains why $\delta_V$ performs poor. Another explanation is that the $r_{ij}$ here are all close to 1/2, but (16) uses 1 or 0 instead; therefore, the solution may be severely biased. Besides $\delta_V$, the other three rules have done very well in this example.

Since $\delta_{HT}$ relies on the approximation $p_i + p_j \approx k/2$, this rule may suffer some losses if the class probabilities are not highly balanced. To examine this point, we consider the following two sets of class probabilities:

(1) We let $k_1 = k/2$ if $k$ is even, and $(k + 1)/2$ if $k$ is odd; then we define $p_1 = 0.95 \times 1.5/k_1$, $p_i = (0.95 - p_1)/(k_1 - 1)$ for $i = 2, \ldots, k_1$, and $p_i = 0.05/(k - k_1)$ for $i = k_1 + 1, \ldots, k$.

(2) If $k = 3$, we define $p_1 = 0.95 \times 1.5/2$, $p_2 = 0.95 - p_1$, and $p_3 = 0.05$. If $k > 3$, we define $p_1 = 0.475$, $p_2 = p_3 = 0.475/2$, and $p_i = 0.05/(k - 3)$ for $i = 4, \ldots, k$.

After setting $p_i$, we define the pairwise comparisons $r_{ij}$ as in (17)-(18). Both experiments are repeated for 1,000 times. The accuracy rates are shown in Figures 1(b) and 1(c). In both scenarios, $p_i$ are not balanced. As expected, $\delta_{HT}$ is quite sensitive to the imbalance of $p_i$. The situation is much worse in Figure 1(c) because the approximation $p_i + p_j \approx k/2$ is more seriously violated, especially when $k$ is large.

In summary, $\delta_1$ and $\delta_2$ are less sensitive to $p_i$, and their overall performance are fairly stable. All features observed here agree with our analysis in Section 5.

## 6.2 Real Data

In this section we present experimental results on several multi-class problems: segment, satimage, and letter from the Statlog collection [9], USPS [4], and MNIST [7]. All data sets are available at `http://www.csie.ntu.edu.tw/~cjlin/libsvmtools/ t`. Their numbers of classes are 7, 6, 26, 10, and 10, respectively. From thousands of instances in each data, we select 300 and 500 as our training and testing sets.

We consider support vector machines (SVM) with RBF kernel $e^{-\gamma\|x_i - x_j\|^2}$ as the binary classifier. The regularization parameter $C$ and the kernel parameter $\gamma$ are selected by cross-validation. To begin, for each training set, a five-fold cross-validation is conducted on the following points of $(C, \gamma)$: $[2^{-5}, 2^{-3}, \ldots, 2^{15}] \times [2^{-5}, 2^{-3}, \ldots, 2^{15}]$. This is done by modifying LIBSVM [1], a library for SVM. At each $(C, \gamma)$, sequentially four folds are

Table 1: Testing errors (in percentage) by four methods: Each row reports the testing errors based on a pair of the training and testing sets. The mean and std (standard deviation) are from five 5-fold cross-validation procedures to select the best $(C, \gamma)$.

| Dataset | k | $\delta_{HT}$ | | $\delta_1$ | | $\delta_2$ | | $\delta_V$ | |
|---|---|---|---|---|---|---|---|---|---|
| | | mean | std | mean | std | mean | std | mean | std |
| satimage | 6 | 14.080 | 1.306 | 14.600 | 0.938 | 14.760 | 0.784 | 15.400 | 0.219 |
| | | 12.960 | 0.320 | 13.400 | 0.400 | 13.400 | 0.400 | 13.360 | 0.080 |
| | | 14.520 | 0.968 | 14.760 | 1.637 | 13.880 | 0.392 | 14.080 | 0.240 |
| | | 12.400 | 0.000 | 12.200 | 0.000 | 12.640 | 0.294 | 12.680 | 1.114 |
| | | 16.160 | 0.294 | 16.400 | 0.379 | 16.120 | 0.299 | 16.160 | 0.344 |
| segment | 7 | 9.960 | 0.480 | 9.480 | 0.240 | 9.000 | 0.400 | 8.880 | 0.271 |
| | | 6.040 | 0.528 | 6.280 | 0.299 | 6.200 | 0.456 | 6.760 | 0.445 |
| | | 6.600 | 0.000 | 6.680 | 0.349 | 6.920 | 0.271 | 7.160 | 0.196 |
| | | 5.520 | 0.466 | 5.200 | 0.420 | 5.400 | 0.580 | 5.480 | 0.588 |
| | | 7.440 | 0.625 | 8.160 | 0.637 | 8.040 | 0.408 | 7.840 | 0.344 |
| USPS | 10 | 14.840 | 0.388 | 13.520 | 0.560 | 12.760 | 0.233 | 12.520 | 0.160 |
| | | 12.080 | 0.560 | 11.440 | 0.625 | 11.600 | 1.081 | 11.440 | 0.991 |
| | | 10.640 | 0.933 | 10.000 | 0.657 | 9.920 | 0.483 | 10.320 | 0.744 |
| | | 12.320 | 0.845 | 11.960 | 1.031 | 11.560 | 0.784 | 11.840 | 1.248 |
| | | 13.400 | 0.310 | 12.640 | 0.080 | 12.920 | 0.299 | 12.520 | 0.917 |
| MNIST | 10 | 17.400 | 0.000 | 16.560 | 0.080 | 15.760 | 0.196 | 15.960 | 0.463 |
| | | 15.200 | 0.400 | 14.600 | 0.000 | 13.720 | 0.588 | 12.360 | 0.196 |
| | | 17.320 | 1.608 | 14.280 | 0.560 | 13.400 | 0.657 | 13.760 | 0.794 |
| | | 14.720 | 0.449 | 14.160 | 0.196 | 13.360 | 0.686 | 13.520 | 0.325 |
| | | 12.560 | 0.294 | 12.600 | 0.000 | 13.080 | 0.560 | 12.440 | 0.233 |
| letter | 26 | 39.880 | 1.412 | 37.160 | 1.106 | 34.560 | 2.144 | 33.480 | 0.325 |
| | | 41.640 | 0.463 | 39.400 | 0.769 | 35.920 | 1.389 | 33.440 | 1.061 |
| | | 41.320 | 1.700 | 38.920 | 0.854 | 35.800 | 1.453 | 35.000 | 1.066 |
| | | 35.240 | 1.439 | 32.920 | 1.121 | 29.240 | 1.335 | 27.400 | 1.117 |
| | | 43.240 | 0.637 | 40.360 | 1.472 | 36.960 | 1.741 | 34.520 | 1.001 |

used as the training set while one fold as the validation set. The training of the four folds consists of $k(k-1)/2$ binary SVMs. For the binary SVM of the $i$th and the $j$th classes, using decision values $\hat{f}$ of training data, we employ an improved implementation [8] of Platt's posterior probabilities [10] to estimate $r_{ij}$:

$$r_{ij} = P(i \mid i \text{ or } j, x) = \frac{1}{1 + e^{A\hat{f}+B}}, \qquad (19)$$

where $A$ and $B$ are estimated by minimizing the negative log-likelihood function.[†]

Then, for each validation instance , we apply the four methods to obtain classification decisions. The error of the five validation sets is thus the cross-validation error at $(C, \gamma)$.

After the cross-validation is done, each rule obtains its best $(C, \gamma)$.[‡] Using these parameters, we train the whole training set to obtain the final model. Next, the same as (19), the decision values from the training data are employed to find $r_{ij}$. Then, testing data are tested using each of the four rules.

Due to the randomness of separating training data into five folds for finding the best $(C, \gamma)$, we repeat the five-fold cross-validation five times and obtain the mean and standard deviation of the testing error. Moreover, as the selection of 300 and 500 training and testing instances from a larger dataset is also random, we generate five of such pairs. In Table 1, each row reports the testing error based on a pair of the training and testing sets. The results show that when the number of classes $k$ is small, the four methods perform similarly; however, for problems with larger $k$, $\delta_{HT}$ is less competitive. In particular, for problem letter which has 26 classes, $\delta_2$ or $\delta_V$ outperforms $\delta_{HT}$ by at least 5%. It seems that for

---

[†][10] suggests to use $\hat{f}$ from the validation instead of the training. However, this requires a further cross-validation on the four-fold data. For simplicity, we directly use $\hat{f}$ from the training.

[‡]If more than one parameter sets return the smallest cross-validation error, we simply choose one with the smallest $C$.

problems here, their characteristics are closer to the setting of Figure 1(c), rather than that of Figure 1(a). All these results agree with the previous findings in Sections 5 and 6.1. Note that in Table 1, some standard deviations are zero. That means the best $(C, \gamma)$ by different cross-validations are all the same. Overall, the variation on parameter selection due to the randomness of cross-validation is not large.

## 7    Discussions and Conclusions

As the minimization of the KL distance is a well known criterion, some may wonder why the performance of $\delta_{HT}$ is not quite satisfactory in some of the examples. One possible explanation is that here KL distance is derived under the assumptions that $n_{ij}r_{ij} \sim$ $\mathrm{Bin}(n_{ij}, \mu_{ij})$ and $r_{ij}$ are independent; however, as pointed out in [3], neither of the assumptions holds in the classification problem.

In conclusion, we have provided two methods which are shown to be more stable than both $\delta_{HT}$ and $\delta_V$. In addition, the two proposed approaches require only solutions of linear systems instead of a nonlinear one in [3].

The authors thank S. Sathiya Keerthi for helpful comments.

## Footnotes

*For $I$ to be well defined, we consider $n_{ij} \neq r_{ji}$, which is generally true. In addition, if there is an $i$ for which $\sum_{j:j\neq i} I_{\{r_{ji}>r_{ij}\}} = 0$, an optimal solution of (16) is $p_i = 1$, and $p_j = 0, \forall j \neq i$. The resulting decision is the same as that of (1).

## References

[1] C.-C. Chang and C.-J. Lin. *LIBSVM: a library for support vector machines*, 2001. Software available at `http://www.csie.ntu.edu.tw/~cjlin/libsvm`.

[2] J. Friedman. Another approach to polychotomous classification. Technical report, Department of Statistics, Stanford University, 1996. Available at `http://www-stat.stanford.edu/reports/friedman/poly.ps.Z`.

[3] T. Hastie and R. Tibshirani. Classification by pairwise coupling. *The Annals of Statistics*, 26(1):451–471, 1998.

[4] J. J. Hull. A database for handwritten text recognition research. *IEEE Transactions on Pattern Analysis and Machine Intelligence*, 16(5):550–554, May 1994.

[5] D. R. Hunter. MM algorithms for generalized Bradley-Terry models. *The Annals of Statistics*, 2004. To appear.

[6] S. Knerr, L. Personnaz, and G. Dreyfus. Single-layer learning revisited: a stepwise procedure for building and training a neural network. In J. Fogelman, editor, *Neurocomputing: Algorithms, Architectures and Applications*. Springer-Verlag, 1990.

[7] Y. LeCun, L. Bottou, Y. Bengio, and P. Haffner. Gradient-based learning applied to document recognition. *Proceedings of the IEEE*, 86(11):2278–2324, November 1998. MNIST database available at `http://yann.lecun.com/exdb/mnist/`.

[8] H.-T. Lin, C.-J. Lin, and R. C. Weng. A note on Platt's probabilistic outputs for support vector machines. Technical report, Department of Computer Science and Information Engineering, National Taiwan University, 2003.

[9] D. Michie, D. J. Spiegelhalter, and C. C. Taylor. *Machine Learning, Neural and Statistical Classification*. Prentice Hall, Englewood Cliffs, N.J., 1994. Data available at `http://www.ncc.up.pt/liacc/ML/statlog/datasets.html`.

[10] J. Platt. Probabilistic outputs for support vector machines and comparison to regularized likelihood methods. In A. Smola, P. Bartlett, B. Schölkopf, and D. Schuurmans, editors, *Advances in Large Margin Classifiers*, Cambridge, MA, 2000. MIT Press.

[11] D. Price, S. Knerr, L. Personnaz, and G. Dreyfus. Pairwise nerual network classifiers with probabilistic outputs. In G. Tesauro, D. Touretzky, and T. Leen, editors, *Neural Information Processing Systems*, volume 7, pages 1109–1116. The MIT Press, 1995.

[12] P. Refregier and F. Vallet. Probabilistic approach for multiclass classification with neural networks. In *Proceedings of International Conference on Artificial Networks*, pages 1003–1007, 1991.
